# Silicon Retina with Adaptive Filtering Properties

**Shih-Chii Liu**
Computation and Neural Systems
136-93 California Institute of Technology
Pasadena, CA 91125
shih@pcmp.caltech.edu

## Abstract

This paper describes a small, compact circuit that captures the temporal and adaptation properties both of the photoreceptor and of the laminar layers of the fly. This circuit uses only six transistors and two capacitors. It is operated in the subthreshold domain. The circuit maintains a high transient gain by using adaptation to the background intensity as a form of gain control. The adaptation time constant of the circuit can be controlled via an external bias. Its temporal filtering properties change with the background intensity or signal-to-noise conditions. The frequency response of the circuit shows that in the frequency range of 1 to 100 Hz, the circuit response goes from highpass filtering under high light levels to lowpass filtering under low light levels (i.e., when the signal-to-noise ratio is low). A chip with 20×20 pixels has been fabricated in 1.2$\mu$m ORBIT CMOS nwell technology.

## 1 BACKGROUND

The first two layers in the fly visual system are the retina layer and the laminar layer. The photoreceptors in the retina synapse onto the monopolar cells in the laminar layer. The photoreceptors adapt to the background intensity, and use this adaptation as a form of gain control in maintaining a high response to transient signals. The laminar layer performs bandpass filtering under high background intensities, and reverts to lowpass filtering in the case of low background intensities where the signal-to-noise (S/N) ratio is low. This adaptive filtering response in the temporal domain is analogous to the spatial center-surround response of the bipolar cells in the vertebrate retina.

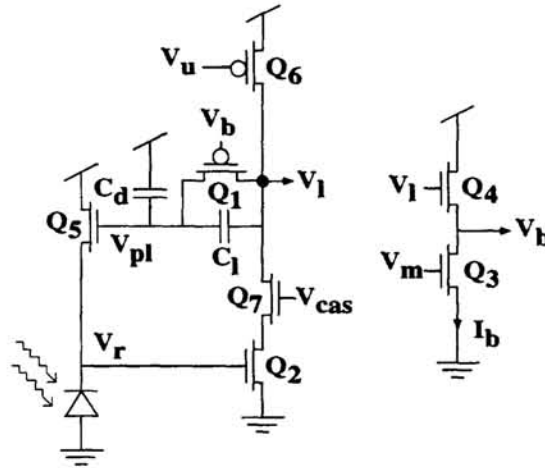

Figure 1: Circuit diagram of retino-laminar circuit. The feedback consists of a resistor implemented by a pFET transistor, $Q_1$. The conductance of the resistor is controlled by the external bias, $V_m$.

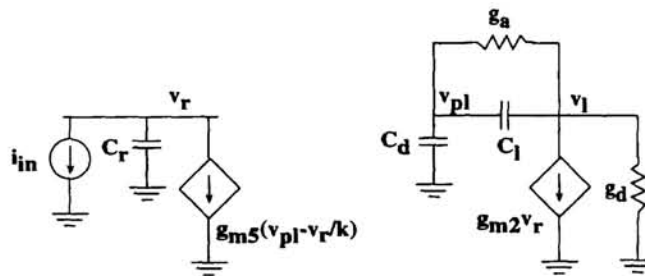

Figure 2: Small signal model of the circuit shown in Figure 1. $C_r$ is the parasitic capacitance at the node, $V_r$.

The Delbrück silicon receptor circuit (Delbrück, 1994) modeled closely the step responses and the adaptation responses of the biological receptors. By including two additional transistors, the **retino-laminar (RL) circuit** described here captures the properties of both the photoreceptor layer (i.e., the adaptation properties and phototransduction) and the cells in the laminar layer (i.e., the adaptive filtering). The time constant of the circuit is controllable via an external bias, and the adaptation behavior of the circuit over different background intensities is more symmetrical than that of Delbrück's photoreceptor circuit.

## 2   CIRCUIT DESCRIPTION

The RL circuit which has the basic form of Delbrück's receptor circuit is shown in Figure 1. I have replaced the adaptive element in his receptor circuit by a nonlinear resistor consisting of a pFET transistor, $Q_1$. The implementation of a floating, voltage-controlled resistor has been described earlier by (Banu and Tsividis, 1982). The bias for $Q_1$, $V_b$, is generated by $Q_3$ and $Q_4$. The conductance of $Q_1$ is determined by the output voltage, $V_1$, and an external bias, $V_m$. We give a brief description of the circuit operation here; details are described in (Delbrück, 1994). The receptor node, $V_r$, is clamped to the voltage needed to sink the current sourced

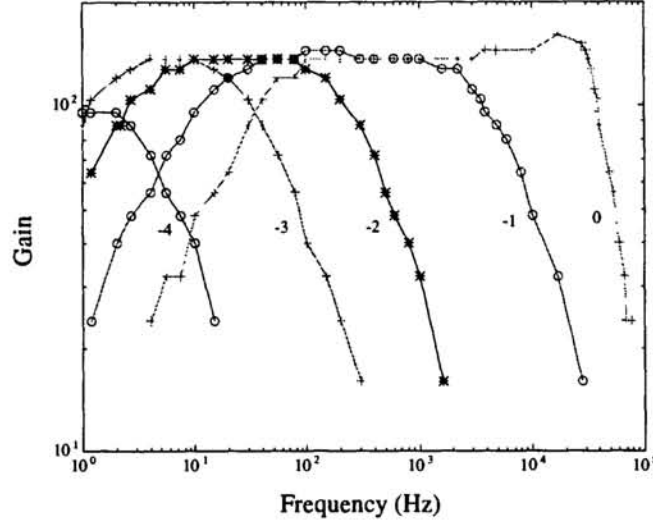

Figure 3: Frequency plot of the RL circuit over five decades of background intensity. The number next to each curve corresponds to the log intensity of the mean value; 0 log corresponds to the intensity of a red LED. The plot shows that, in the range of 1 to 100 Hz, the circuit is a bandpass filter at high light levels, and reduces to a lowpass filter at low light levels.

by $Q_6$, which is biased by an external voltage, $V_u$. Changes in the photocurrent are amplified by the transistors, $Q_2$ and $Q_6$, resulting in a change in $V_l$. This change in $V_l$ is capacitively coupled through the capacitive divider, consisting of $C_1$ and $C_d$, into $V_{pl}$, so that $Q_5$ supplies the extra increase in photocurrent.

The feedback transistor, $Q_5$, is operated in subthreshold so that $V_r$ and $V_l$ is logarithmic in the photocurrent. A large change in the photocurrent resulting from a change in the background intensity, leads to a large change in the circuit output, $V_l$. Significant current then flows through $Q_1$, thus charging or discharging $V_{pl}$.

## 3   PROPERTIES OF RL CIRCUIT

The temporal responses and adaptation properties of this circuit are expounded in the following sections. In Section 3.1, we solve for the transfer function of the circuit and in Section 3.2, we describe the dependence of the conductance of $Q_1$ on the background intensity. In Sections 3.3 and 3.4, we describe the temporal responses of this circuit, and compare the adaptation response of RL circuit with that of Delbrück's circuit.

### 3.1   TRANSFER FUNCTION

We can solve for the transfer function of the RL circuit in Figure 1 by writing the KCL equations of the small-signal model shown in Figure 2. The transfer function, $\frac{v_l}{i_{in}}$, is given by:

$$\frac{v_l}{i_{in}} = \frac{1}{g_{m5}}\left[\frac{\frac{s(\tau_{ld}+\tau_l)+g_a/g_{m2}}{s\tau_l+g_a/g_{m2}}}{(s\tau_r + 1/\kappa)(1/A_{amp} + s\tau_{ld} + \frac{s\tau_{ld}}{A_{amp}(s\tau_l+g_a/g_{m2})}) + 1}\right], \quad (1)$$

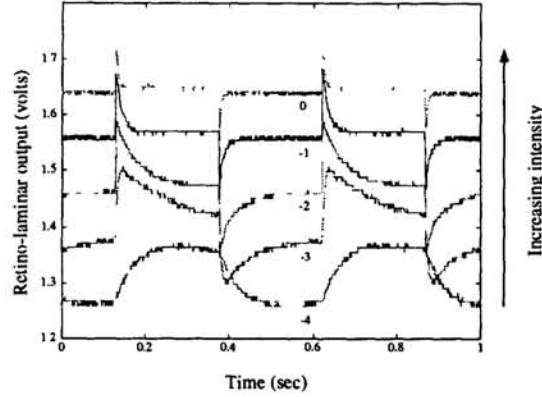

Figure 4: Temporal responses of the circuit over five decades of background intensity. The input stimulus is a square-wave-modulated red LED of contrast 0.15. The circuit acts as a highpass filter (that is, a differentiator) at high intensities, and as a lowpass filter as the intensity drops.

where $A_{amp} = \frac{g_{m2}}{g_d}$, $g_m$ is the transconductance, and $g_d$ is the output conductance of a transistor. We define the time constants, $\tau_l$, $\tau_r$, and $\tau_{ld}$, as follows:

$$\tau_l = \frac{C_l}{g_{m2}}; \tau_r = \frac{C_r}{g_{m5}}; \tau_{ld} = \frac{C_d}{g_{m2}},$$

where $g_a$ is the output conductance of $Q_1$ and $C_r$ is the parasitic capacitance at the node, $V_r$.

The frequency response curves in Figure 3 are measured from the fabricated circuit over five decades of background intensity. We obtain the curves by using a sine-wave–modulated red LED source. The number next to each curve is the log intensity of the mean value; 0 log is the intensity of a red LED. We obtain the remaining curves by interposing neutral density filters between the LED source and the chip. Figure 3 shows that, in the range of 1 to 100 Hz, the circuit is a bandpass filter at high light levels, and reduces to a lowpass filter at low light levels. For each frequency curve, the gain is flat in the middle, and is given by $A_{\mathrm{cl}} = \frac{C_l + C_d}{C_l}$. The cutoff frequencies change with the background intensity; this change is analyzed in Section 3.2.

## 3.2 DEPENDENCE OF CIRCUIT'S TIME CONSTANT ON BACKGROUND INTENSITY

The cutoff frequencies of the circuit depend on the conductance, $g_a$, of $Q_1$. Here, we analyze the dependence of $g_a$ on the background intensity. Since $Q_1$ is operated in subthreshold, the conductance depends on the current flowing through $Q_1$. The I–V relationship for $Q_1$ can be written as

$$I = 2I_{op}\Big(\frac{I_b}{I_{on}}\Big)^{\kappa_p} e^{(1-\kappa_n\kappa_p)\overline{V}} e^{-\kappa_n\kappa_p\Delta V/2} \sinh(\Delta V/2) \tag{2}$$

$$= I_\alpha I_{\mathrm{ph}}^{(1-\kappa_n\kappa_p)/\kappa_n} e^{(1-2\kappa_n\kappa_p)\Delta V/2} \sinh(\Delta V/2) \tag{3}$$

where $\overline{V} = \frac{V_l + V_{pl}}{2}$, $\Delta V = V_l - V_{pl}$, $I_{\mathrm{ph}}$ is the photocurrent, and $I_\alpha = 2I_{op}\big(\frac{I_b}{I_{on}}\big)^{\kappa_p}\big(\frac{e^{V_r}}{I_{on}}\big)^{(1-\kappa_n\kappa_p)/\kappa_n}$. The exponential relationship for Equations 2 and 3 is for a FET transistor operating in subthreshold, where $I_{op}$ is the quiescent leakage current of the transistor, and $\kappa$ is the effectiveness of the gate in controlling

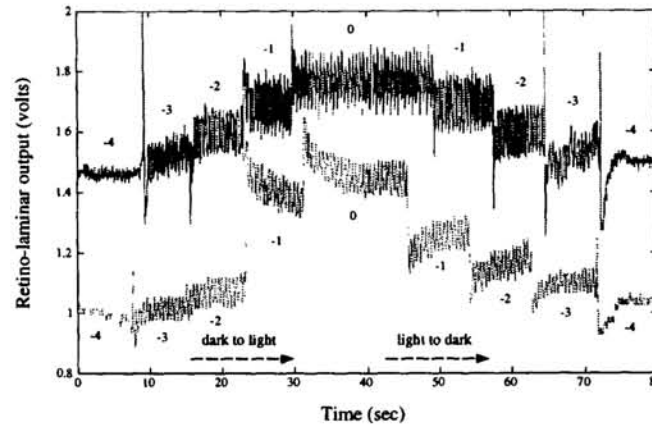

Figure 5: Plots of adaptation responses of the RL circuit and of Delbrück's circuit. The input stimulus is a red LED driven by a square wave of contrast 0.18. The bottom curve corresponding to Delbrück's receptor has been shifted down so that we can compare the two curves. The adaptation response of the RL circuit is more symmetrical than that of Delbrück's circuit when the circuit goes from dark to light conditions and back.

the surface potential of the channel of the transistor. Equation 3 shows that $g_a$ is proportional to the photocurrent, $I_{ph}$, hence, the background intensity. A more intuitive way of understanding how $g_a$ changes with $I_{ph}$ is that the change in $V_b$ with a fixed change in the output, $V_l$, depends on the output level of $V_l$. The change in $V_b$ is larger for a higher DC output, $V_l$, because of the increased body effect at $Q_4$ due to its higher source voltage. The larger change in $V_b$ leads to an increase in the conductance, $g_a$.

As $I_{ph}$ increases, $g_a$ increases, so the cutoff frequencies shift to the right, as seen in Figure 3. If we compare both the "0" curve and the "-1" curve, we can see that the cutoff frequencies are approximately different by a factor of 10. Thus, the exponent of $I_{ph}$, $(1 - \kappa_n\kappa_p)/\kappa_n \approx 1$. Since the $\kappa$ values change with the current through the transistor, the exponent also changes. The different values of the exponent with $I_{ph}$ can be seen from the different amounts of shifts in the cutoff frequencies of the curves.

## 3.3 TEMPORAL RESPONSES

The adaptive temporal filtering of the circuit over five decades of background intensity can also be observed from the step response of the RL circuit to a square-wave-modulated LED of contrast 0.15, as shown in Figure 4. The data in Figure 4 show that the time constant of the circuit increases as the light level decreases. The temporal responses observed in these circuits are comparable to the contrast responses recorded from the LMCs by Juusola and colleagues (Juusola et al., 1995).

## 3.4 ADAPTATION PROPERTIES

The RL circuit also differs from Delbrück's circuit in that the adaptation time constant can be set by an external bias. In the Delbrück circuit, the adaptation time constant is predetermined at the design phase and by process parameters. In Figure 5, we compare the adaptation properties of the RL circuit with those of Delbrück's

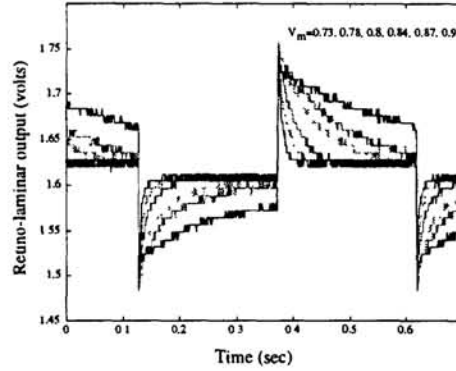

Figure 6: Step response of the RL circuit for different values of $V_m$. The input stimulus is a square-wave-modulated red LED source. The value of $V_m$ was varied from 0.73 to 0.9 V. The curve with the longest time constant of decay corresponds to the lowest value of $V_m$.

circuit. The input stimulus consists of a square-wave-modulated LED source with a contrast of about 0.18. We take the circuit from dark to light conditions, and back again, by using neutral density filters. The top curve corresponds to the response from the RL circuit, and the bottom curve corresponds to the response from the Delbrück circuit. The RL circuit adapts symmetrically, when it goes from light to dark conditions and back. In contrast, Delbrück's circuit shows an asymmetrical adaptative behavior; it adapts more slowly when it goes from dark to light conditions.

The adaptation time constant of the RL circuit depends on the conductance, $g_a$, and the capacitors, $C_l$ and $C_d$. From Equation 3, we see that $g_a$ is dependent on $I_b$ which is set by the bias, $V_m$. Hence, we can change the adaptation time constant by varying $V_m$. The dependence of the time constant on $V_m$ is further demonstrated by recording the step response of the circuit to a LED source of contrast 0.15 for various values of $V_m$. The output data is shown in Figure 6 for five different values of $V_m$. The time constant of the circuit decreases as $V_m$ increases.

A chip consisting of $20 \times 20$ pixels was fabricated in $1.2\mu m$ ORBIT CMOS nwell technology. An input stimulus consisting of a rotating flywheel, with black strips on a white background, was initially presented to the imager. The flywheel was then stopped, and the response of the chip was recorded one sec after the motion was ceased. I repeated the experiment for two adaptation time constants by changing the value of $V_m$. Figure 7a shows the output of the chip with the longer adaptation time constant. We see that the image is still present, whereas the image in Figure 7b has almost faded away; that is, the chip has adapted away the stationary image.

## 4   CONCLUSIONS

I have described a small circuit captures the temporal and adaptation properties of both the photoreceptor and the laminar layers in the fly retina. By adapting to the background intensity, the circuit maintains a high transient gain. The temporal behavior of the circuit also changes with the background intensity, such that, at high S/N ratios, the circuit acts as a highpass filter and, at low S/N ratios, the circuit acts as a lowpass filter to average out the noise. The circuit uses only six transistors and two capacitors and is compact. The adaptation time constant of the

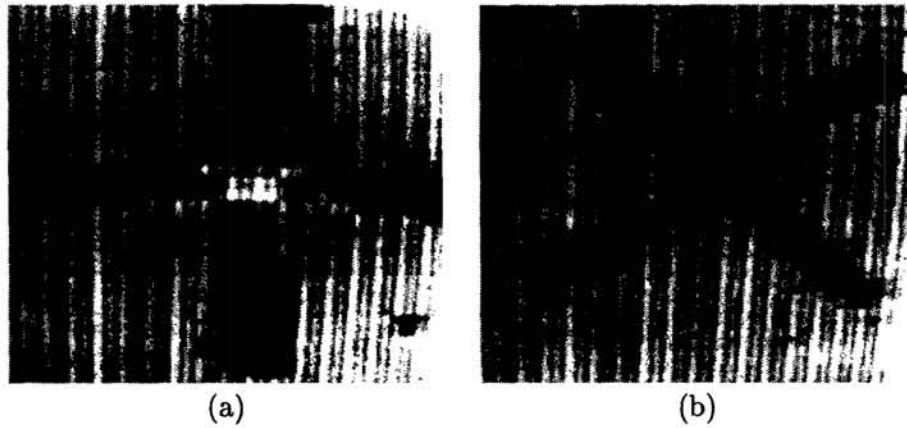

(a)                                    (b)

Figure 7: Adaptation results from a two-dimensional array of 20 × 20 pixels. The output of the array was recorded one sec after cessation of the pattern motion. The experiment was repeated for two different adaptation time constants. Figure (a) corresponds to the longer adaptation time constant. The image is still present, whereas the image in Figure (b) has almost faded away.

circuit can be controlled via an external bias.

## Acknowledgments

I thank Bradley A. Minch for discussions of this work, Carver Mead for supporting this work, and the MOSIS foundation for fabricating this circuit. I also thank Lyn Dupre for editing this document. This work was supported in part by the Office of Naval Research, by DARPA, and by the Beckman Foundation.

## References

T. Delbrück, "Analog VLSI phototransduction by continous-time, adaptive, logarithmic photoreceptor circuits," *CNS Memo No.30*, California Institute of Technology, Pasadena, CA, 1994.

M. Banu and Y. Tsividis, "Floating voltage-controlled resistors in CMOS technology," *Electronics Letters*, **18:15**, pp. 678–679, 1982.

M. Juusola, R.O. Uusitola, and M. Weckstrom, " Transfer of graded potentials at the photoreceptor-interneuron synapse," *J. of General Physiology*, **105**, pp. 115–148, 1995.